# Monte Carlo Sampling for Regret Minimization in Extensive Games

**Marc Lanctot**
Department of Computing Science
University of Alberta
Edmonton, Alberta, Canada T6G 2E8
`lanctot@ualberta.ca`

**Kevin Waugh**
School of Computer Science
Carnegie Mellon University
Pittsburgh PA 15213-3891
`waugh@cs.cmu.edu`

**Martin Zinkevich**
Yahoo! Research
Santa Clara, CA, USA 95054
`maz@yahoo-inc.com`

**Michael Bowling**
Department of Computing Science
University of Alberta
Edmonton, Alberta, Canada T6G 2E8
`bowling@cs.ualberta.ca`

## Abstract

Sequential decision-making with multiple agents and imperfect information is commonly modeled as an extensive game. One efficient method for computing Nash equilibria in large, zero-sum, imperfect information games is counterfactual regret minimization (CFR). In the domain of poker, CFR has proven effective, particularly when using a domain-specific augmentation involving chance outcome sampling. In this paper, we describe a general family of domain-independent CFR sample-based algorithms called Monte Carlo counterfactual regret minimization (MCCFR) of which the original and poker-specific versions are special cases. We start by showing that MCCFR performs the same regret updates as CFR on expectation. Then, we introduce two sampling schemes: *outcome sampling* and *external sampling*, showing that both have bounded overall regret with high probability. Thus, they can compute an approximate equilibrium using self-play. Finally, we prove a new tighter bound on the regret for the original CFR algorithm and relate this new bound to MCCFR's bounds. We show empirically that, although the sample-based algorithms require more iterations, their lower cost per iteration can lead to dramatically faster convergence in various games.

## 1 Introduction

Extensive games are a powerful model of sequential decision-making with imperfect information, subsuming finite-horizon MDPs, finite-horizon POMDPs, and perfect information games. The past few years have seen dramatic algorithmic improvements in solving, *i.e.,* finding an approximate Nash equilibrium, in two-player, zero-sum extensive games. Multiple techniques [1, 2] now exist for solving games with up to $10^{12}$ game states, which is about four orders of magnitude larger than the previous state-of-the-art of using sequence-form linear programs [3].

*Counterfactual regret minimization* (CFR) [1] is one such recent technique that exploits the fact that the time-averaged strategy profile of regret minimizing algorithms converges to a Nash equilibrium. The key insight is the fact that minimizing per-information set counterfactual regret results in minimizing overall regret. However, the vanilla form presented by Zinkevich and colleagues requires the entire game tree to be traversed on each iteration. It is possible to avoid a full game-tree traversal. In their accompanying technical report, Zinkevich and colleagues discuss a *poker-specific* CFR

variant that samples chance outcomes on each iteration [4]. They claim that the per-iteration cost reduction far exceeds the additional number of iterations required, and all of their empirical studies focus on this variant. The sampling variant and its derived bound are limited to poker-like games where chance plays a prominent role in the size of the games. This limits the practicality of CFR minimization outside of its initial application of poker or moderately sized games. An additional disadvantage of CFR is that it requires the opponent's policy to be known, which makes it unsuitable for online regret minimization in an extensive game. Online regret minimization in extensive games is possible using online convex programming techniques, such as Lagrangian Hedging [5], but these techniques can require costly optimization routines at every time step.

In this paper, we present a general framework for sampling in counterfactual regret minimization. We define a family of Monte Carlo CFR minimizing algorithms (MCCFR), that differ in how they sample the game tree on each iteration. Zinkevich's vanilla CFR and a generalization of their chance-sampled CFR are both members of this family. We then introduce two additional members of this family: *outcome-sampling*, where only a single playing of the game is sampled on each iteration; and *external-sampling*, which samples chance nodes and the opponent's actions. We show that under a reasonable sampling strategy, any member of this family minimizes overall regret, and so can be used for equilibrium computation. Additionally, external-sampling is proven to require only a constant-factor increase in iterations yet achieves an order reduction in the cost per iteration, thus resulting an asymptotic improvement in equilibrium computation time. Furthermore, since outcome-sampling does not need knowledge of the opponent's strategy beyond samples of play from the strategy, we describe how it can be used for online regret minimization. We then evaluate these algorithms empirically by using them to compute approximate equilibria in a variety of games.

## 2   Background

An extensive game is a general model of sequential decision-making with imperfect information. As with perfect information games (such as Chess or Checkers), extensive games consist primarily of a game tree: each non-terminal node has an associated player (possibly chance) that makes the decision at that node, and each terminal node has associated utilities for the players. Additionally, game states are partitioned into information sets where a player cannot distinguish between two states in the same information set. The players, therefore, must choose actions with the same distribution at each state in the same information set. We now define an extensive game formally, introducing the notation we use throughout the paper.

**Definition 1** *[6, p. 200] a finite extensive game with imperfect information has the following components:*

- *A finite set $N$ of **players**. A finite set $H$ of sequences, the possible **histories** of actions, such that the empty sequence is in $H$ and every prefix of a sequence in $H$ is also in $H$. Define $h \sqsubseteq h'$ to mean $h$ is a prefix of $h'$. $Z \subseteq H$ are the terminal histories (those which are not a prefix of any other sequences). $A(h) = \{a : ha \in H\}$ are the actions available after a non-terminal history, $h \in H \setminus Z$.*
- *A function $P$ that assigns to each non-terminal history a member of $N \cup \{c\}$. $P$ is the **player function**. $P(h)$ is the player who takes an action after the history $h$. If $P(h) = c$ then chance determines the action taken after history $h$.*
- *For each player $i \in N \cup \{c\}$ a partition $\mathcal{I}_i$ of $\{h \in H : P(h) = i\}$ with the property that $A(h) = A(h')$ whenever $h$ and $h'$ are in the same member of the partition. For $I_i \in \mathcal{I}_i$ we denote by $A(I_i)$ the set $A(h)$ and by $P(I_i)$ the player $P(h)$ for any $h \in I_i$. $\mathcal{I}_i$ is the **information partition** of player $i$; a set $I_i \in \mathcal{I}_i$ is an **information set** of player $i$.*
- *A function $f_c$ that associates with every information set $I$ where $P(I) = c$ a probability measure $f_c(\cdot|I)$ on $A(h)$ ($f_c(a|I)$ is the probability that $a$ occurs given some $h \in I$), where each such probability measure is independent of every other such measure.[1]*

- *For each player $i \in N$ a utility function $u_i$ from the terminal states $Z$ to the reals $\mathbf{R}$. If $N = \{1, 2\}$ and $u_1 = -u_2$, it is a **zero-sum extensive game**. Define $\Delta_{u,i} = \max_z u_i(z) - \min_z u_i(z)$ to be the range of utilities to player $i$.*

In this paper, we will only concern ourselves with two-player, zero-sum extensive games. Furthermore, we will assume **perfect recall**, a restriction on the information partitions such that a player can always distinguish between game states where they previously took a different action or were previously in a different information set.

## 2.1 Strategies and Equilibria

A **strategy of player $i$**, $\sigma_i$, in an extensive game is a function that assigns a distribution over $A(I_i)$ to each $I_i \in \mathcal{I}_i$. We denote $\Sigma_i$ as the set of all strategies for player $i$. A **strategy profile**, $\sigma$, consists of a strategy for each player, $\sigma_1, \dots, \sigma_n$. We let $\sigma_{-i}$ refer to the strategies in $\sigma$ excluding $\sigma_i$.

Let $\pi^\sigma(h)$ be the probability of history $h$ occurring if all players choose actions according to $\sigma$. We can decompose $\pi^\sigma(h) = \Pi_{i \in N \cup \{c\}} \pi_i^\sigma(h)$ into each player's contribution to this probability. Here, $\pi_i^\sigma(h)$ is the contribution to this probability from player $i$ when playing according to $\sigma$. Let $\pi_{-i}^\sigma(h)$ be the product of all players' contribution (including chance) except that of player $i$. For $I \subseteq H$, define $\pi^\sigma(I) = \sum_{h \in I} \pi^\sigma(h)$, as the probability of reaching a particular information set given all players play according to $\sigma$, with $\pi_i^\sigma(I)$ and $\pi_{-i}^\sigma(I)$ defined similarly. Finally, let $\pi^\sigma(h, z) = \pi^\sigma(z)/\pi^\sigma(h)$ if $h \sqsubseteq z$, and zero otherwise. Let $\pi_i^\sigma(h, z)$ and $\pi_{-i}^\sigma(h, z)$ be defined similarly. Using this notation, we can define the expected payoff for player $i$ as $u_i(\sigma) = \sum_{h \in Z} u_i(h) \pi^\sigma(h)$.

Given a strategy profile, $\sigma$, we define a player's **best response** as a strategy that maximizes their expected payoff assuming all other players play according to $\sigma$. The **best-response value** for player $i$ is the value of that strategy, $b_i(\sigma_{-i}) = \max_{\sigma_i' \in \Sigma_i} u_i(\sigma_i', \sigma_{-i})$. An **$\epsilon$-Nash equilibrium** is an approximation of a Nash equilibrium; it is a strategy profile $\sigma$ that satisfies

$$\forall i \in N \quad u_i(\sigma) + \epsilon \geq \max_{\sigma_i' \in \Sigma_i} u_i(\sigma_i', \sigma_{-i}) \tag{1}$$

If $\epsilon = 0$ then $\sigma$ is a **Nash Equilibrium**: no player has any incentive to deviate as they are all playing best responses. If a game is two-player and zero-sum, we can use **exploitability** as a metric for determining how close $\sigma$ is to an equilibrium, $\epsilon_\sigma = b_1(\sigma_2) + b_2(\sigma_1)$.

## 2.2 Counterfactual Regret Minimization

Regret is an online learning concept that has triggered a family of powerful learning algorithms. To define this concept, first consider repeatedly playing an extensive game. Let $\sigma_i^t$ be the strategy used by player $i$ on round $t$. The **average overall regret** of player $i$ at time $T$ is:

$$R_i^T = \frac{1}{T} \max_{\sigma_i^* \in \Sigma_i} \sum_{t=1}^{T} \left( u_i(\sigma_i^*, \sigma_{-i}^t) - u_i(\sigma^t) \right) \tag{2}$$

Moreover, define $\bar{\sigma}_i^t$ to be the **average strategy** for player $i$ from time 1 to $T$. In particular, for each information set $I \in \mathcal{I}_i$, for each $a \in A(I)$, define:

$$\bar{\sigma}_i^t(a|I) = \frac{\sum_{t=1}^{T} \pi_i^{\sigma^t}(I) \sigma^t(a|I)}{\sum_{t=1}^{T} \pi_i^{\sigma^t}(I)}. \tag{3}$$

There is a well-known connection between regret, average strategies, and Nash equilibria.

**Theorem 1** *In a zero-sum game, if $R_{i \in \{1,2\}}^T \leq \epsilon$, then $\bar{\sigma}^T$ is a $2\epsilon$ equilibrium.*

An algorithm for selecting $\sigma_i^t$ for player $i$ is regret minimizing if player $i$'s average overall regret (regardless of the sequence $\sigma_{-i}^t$) goes to zero as $t$ goes to infinity. Regret minimizing algorithms in self-play can be used as a technique for computing an approximate Nash equilibrium. Moreover, an algorithm's bounds on the average overall regret bounds the convergence rate of the approximation.

Zinkevich and colleagues [1] used the above approach in their counterfactual regret algorithm (CFR). The basic idea of CFR is that overall regret can be bounded by the sum of positive per-information-set immediate counterfactual regret. Let $I$ be an information set of player $i$. Define $\sigma_{(I \to a)}$ to be

a strategy profile identical to $\sigma$ except that player $i$ always chooses action $a$ from information set $I$. Let $Z_I$ be the subset of all terminal histories where a prefix of the history is in the set $I$; for $z \in Z_I$ let $z[I]$ be that prefix. Since we are restricting ourselves to perfect recall games $z[I]$ is unique. Define **counterfactual value** $v_i(\sigma, I)$ as,

$$v_i(\sigma, I) = \sum_{z \in Z_I} \pi^\sigma_{-i}(z[I])\pi^\sigma(z[I], z)u_i(z). \tag{4}$$

The **immediate counterfactual regret** is then $R^T_{i,\mathrm{imm}}(I) = \max_{a \in A(I)} R^T_{i,\mathrm{imm}}(I, a)$, where

$$R^T_{i,\mathrm{imm}}(I, a) = \frac{1}{T}\sum_{t=1}^{T} \left( v_i(\sigma^t_{(I \to a)}, I) - v_i(\sigma^t, I) \right) \tag{5}$$

Let $x^+ = \max(x, 0)$. The key insight of CFR is the following result.

**Theorem 2** *[1, Theorem 3]*    $R^T_i \leq \sum_{I \in \mathcal{I}_i} R^{T,+}_{i,\mathrm{imm}}(I)$

Using regret-matching[2] the positive per-information set immediate counterfactual regrets can be driven to zero, thus driving average overall regret to zero. This results in an average overall regret bound [1, Theorem 4]: $R^T_i \leq \Delta_{u,i}|\mathcal{I}_i|\sqrt{|A_i|}/\sqrt{T}$, where $|A_i| = \max_{h:P(h)=i}|A(h)|$. We return to this bound, tightening it further, in Section 4.

This result suggests an algorithm for computing equilibria via self-play, which we will refer to as *vanilla CFR*. The idea is to traverse the game tree computing counterfactual values using Equation 4. Given a strategy, these values define regret terms for each player for each of their information sets using Equation 5. These regret values accumulate and determine the strategies at the next iteration using the regret-matching formula. Since both players are regret minimizing, Theorem 1 applies and so computing the strategy profile $\bar{\sigma}^t$ gives us an approximate Nash Equilibrium. Since CFR only needs to store values at each information set, its space requirement is $O(|\mathcal{I}|)$. However, as previously mentioned vanilla CFR requires a complete traversal of the game tree on each iteration, which prohibits its use in many large games. Zinkevich and colleagues [4] made steps to alleviate this concern with a chance-sampled variant of CFR for poker-like games.

## 3   Monte Carlo CFR

The key to our approach is to avoid traversing the entire game tree on each iteration while still having the immediate counterfactual regrets be unchanged *in expectation*. In general, we want to restrict the terminal histories we consider on each iteration. Let $\mathcal{Q} = \{Q_1, \dots, Q_r\}$ be a set of subsets of $Z$, such that their union spans the set $Z$. We will call one of these subsets a **block**. On each iteration we will sample one of these blocks and only consider the terminal histories in that block. Let $q_j > 0$ be the probability of considering block $Q_j$ for the current iteration (where $\sum_{j=1}^{r} q_j = 1$).

Let $q(z) = \sum_{j:z \in Q_j} q_j$, *i.e.*, $q(z)$ is the probability of considering terminal history $z$ on the current iteration. The **sampled counterfactual value** when updating block $j$ is:

$$\tilde{v}_i(\sigma, I|j) = \sum_{z \in Q_j \cap Z_I} \frac{1}{q(z)}u_i(z)\pi^\sigma_{-i}(z[I])\pi^\sigma(z[I], z) \tag{6}$$

Selecting a set $\mathcal{Q}$ along with the sampling probabilities defines a complete sample-based CFR algorithm. Rather than doing full game tree traversals the algorithm samples one of these blocks, and then examines only the terminal histories in that block.

Suppose we choose $\mathcal{Q} = \{Z\}$, *i.e.*, one block containing all terminal histories and $q_1 = 1$. In this case, sampled counterfactual value is equal to counterfactual value, and we have vanilla CFR. Suppose instead we choose each block to include all terminal histories with the same sequence of chance outcomes (where the probability of a chance outcome is independent of players' actions as

in poker-like games). Hence $q_j$ is the product of the probabilities in the sampled sequence of chance outcomes (which cancels with these same probabilities in the definition of counterfactual value) and we have Zinkevich and colleagues' chance-sampled CFR.

Sampled counterfactual value was designed to match counterfactual value on expectation. We show this here, and then use this fact to prove a probabilistic bound on the algorithm's average overall regret in the next section.

**Lemma 1** $E_{j \sim q_j} [\tilde{v}_i(\sigma, I|j)] = v_i(\sigma, I)$

**Proof:**

$$E_{j \sim q_j} [\tilde{v}_i(\sigma, I|j)] = \sum_j q_j \tilde{v}_i(\sigma, I|j) = \sum_j \sum_{z \in Q_j \cap Z_I} \frac{q_j}{q(z)} \pi^\sigma_{-i}(z[I]) \pi^\sigma(z[I], z) u_i(z) \quad (7)$$

$$= \sum_{z \in Z_I} \frac{\sum_{j:z \in Q_j} q_j}{q(z)} \pi^\sigma_{-i}(z[I]) \pi^\sigma(z[I], z) u_i(z) \quad (8)$$

$$= \sum_{z \in Z_I} \pi^\sigma_{-i}(z[I]) \pi^\sigma(z[I], z) u_i(z) = v_i(\sigma, I) \quad (9)$$

Equation 8 follows from the fact that $\mathcal{Q}$ spans $Z$. Equation 9 follows from the definition of $q(z)$. ∎

This results in the following MCCFR algorithm. We sample a block and for each information set that contains a prefix of a terminal history in the block we compute the *sampled immediate counterfactual regrets* of each action, $\tilde{r}(I, a) = \tilde{v}_i(\sigma^t_{(I \to a)}, I) - \tilde{v}_i(\sigma^t, I)$. We accumulate these regrets, and the player's strategy on the next iteration applies the regret-matching algorithm to the accumulated regrets. We now present two specific members of this family, giving details on how the regrets can be updated efficiently.

**Outcome-Sampling MCCFR.** In *outcome-sampling MCCFR* we choose $\mathcal{Q}$ so that each block contains a single terminal history, *i.e.,* $\forall Q \in \mathcal{Q}, |Q| = 1$. On each iteration we sample one terminal history and only update each information set along that history. The sampling probabilities, $q_j$ must specify a distribution over terminal histories. We will specify this distribution using a *sampling profile*, $\sigma'$, so that $q(z) = \pi^{\sigma'}(z)$. Note that any choice of sampling policy will induce a particular distribution over the block probabilities $q(z)$. As long as $\sigma'_i(a|I) > \epsilon$, then there exists a $\delta > 0$ such that $q(z) > \delta$, thus ensuring Equation 6 is well-defined.

The algorithm works by sampling $z$ using policy $\sigma'$, storing $\pi^{\sigma'}(z)$. The single history is then traversed forward (to compute each player's probability of playing to reach each prefix of the history, $\pi^\sigma_i(h)$) and backward (to compute each player's probability of playing the remaining actions of the history, $\pi^\sigma_i(h, z)$). During the backward traversal, the sampled counterfactual regrets at each visited information set are computed (and added to the total regret).

$$\tilde{r}(I, a) = \begin{cases} w_I \cdot (1 - \sigma(a|z[I])) & \text{if } (z[I]a) \sqsubseteq z \\ -w_I \cdot \sigma(a|z[I]) & \text{otherwise} \end{cases}, \text{ where } w_I = \frac{u_i(z) \pi^\sigma_{-i}(z) \pi^\sigma_i(z[I]a, z)}{\pi^{\sigma'}(z)} \quad (10)$$

One advantage of outcome-sampling MCCFR is that if our terminal history is sampled according to the opponent's policy, so $\sigma'_{-i} = \sigma_{-i}$, then the update no longer requires explicit knowledge of $\sigma_{-i}$ as it cancels with the $\sigma'_{-i}$. So, $w_I$ becomes $u_i(z) \pi^\sigma_i(z[I], z) / \pi^{\sigma'}_i(z)$. Therefore, we can use outcome-sampling MCCFR for online regret minimization. We would have to choose our own actions so that $\sigma'_i \approx \sigma^t_i$, but with some exploration to guarantee $q_j \geq \delta > 0$. By balancing the regret caused by exploration with the regret caused by a small $\delta$ (see Section 4 for how MCCFR's bound depends upon $\delta$), we can bound the average overall regret as long as the number of playings $T$ is known in advance. This effectively mimics the approach taking by Exp3 for regret minimization in normal-form games [9]. An alternative form for Equation 10 is recommended for implementation. This and other implementation details can be found in the paper's supplemental material or the appendix of the associated technical report [10].

**External-Sampling MCCFR.** In *external-sampling MCCFR* we sample only the actions of the opponent and chance (those choices external to the player). We have a block $Q_\tau \in \mathcal{Q}$ for each pure strategy of the opponent and chance, *i.e.*,, for each deterministic mapping $\tau$ from $I \in \mathcal{I}_c \cup \mathcal{I}_{N \setminus \{i\}}$ to $A(I)$. The block probabilities are assigned based on the distributions $f_c$ and $\sigma_{-i}$, so $q_\tau = \prod_{I \in \mathcal{I}_c} f_c(\tau(I)|I) \prod_{I \in \mathcal{I}_{N \setminus \{i\}}} \sigma_{-i}(\tau(I)|I)$. The block $Q_\tau$ then contains all terminal histories $z$ consistent with $\tau$, that is if $ha$ is a prefix of $z$ with $h \in I$ for some $I \in \mathcal{I}_{-i}$ then $\tau(I) = a$. In practice, we will not actually sample $\tau$ but rather sample the individual actions that make up $\tau$ only as needed. The key insight is that these block probabilities result in $q(z) = \pi^\sigma_{-i}(z)$. The algorithm iterates over $i \in N$ and for each doing a post-order depth-first traversal of the game tree, sampling actions at each history $h$ where $P(h) \neq i$ (storing these choices so the same actions are sampled at all $h$ in the same information set). Due to perfect recall it can never visit more than one history from the same information set during this traversal. For each such visited information set the sampled counterfactual regrets are computed (and added to the total regrets).

$$\tilde{r}(I, a) = (1 - \sigma(a|I)) \sum_{z \in Q \cap Z_I} u_i(z) \pi^\sigma_i(z[I]a, z) \tag{11}$$

Note that the summation can be easily computed during the traversal by always maintaining a weighted sum of the utilities of all terminal histories rooted at the current history.

## 4 Theoretical Analysis

We now present regret bounds for members of the MCCFR family, starting with an improved bound for vanilla CFR that depends more explicitly on the exact structure of the extensive game. Let $\vec{a}_i$ be a subsequence of a history such that it contains only player $i$'s actions in that history, and let $\vec{A}_i$ be the set of all such player $i$ action subsequences. Let $\mathcal{I}_i(\vec{a}_i)$ be the set of all information sets where player $i$'s action sequence up to that information set is $\vec{a}_i$. Define the $M$-value for player $i$ of the game to be $M_i = \sum_{\vec{a}_i \in \vec{A}_i} \sqrt{|\mathcal{I}_i(\vec{a})|}$. Note that $\sqrt{|\mathcal{I}_i|} \leq M_i \leq |\mathcal{I}_i|$ with both sides of this bound being realized by some game. We can strengthen vanilla CFR's regret bound using this constant, which also appears in the bounds for the MCCFR variants.

**Theorem 3** *When using vanilla CFR for player $i$, $R_i^T \leq \Delta_{u,i} M_i \sqrt{|A_i|}/\sqrt{T}$.*

We now turn our attention to the MCCFR family of algorithms, for which we can provide probabilistic regret bounds. We begin with the most exciting result: showing that external-sampling requires only a constant factor more iterations than vanilla CFR (where the constant depends on the desired confidence in the bound).

**Theorem 4** *For any $p \in (0,1]$, when using external-sampling MCCFR, with probability at least $1 - p$, average overall regret is bounded by, $R_i^T \leq \left(1 + \frac{\sqrt{2}}{\sqrt{p}}\right) \Delta_{u,i} M_i \sqrt{|A_i|}/\sqrt{T}$.*

Although requiring the same order of iterations, note that external-sampling need only traverse a fraction of the tree on each iteration. For balanced games where players make roughly equal numbers of decisions, the iteration cost of external-sampling is $O(\sqrt{|H|})$, while vanilla CFR is $O(|H|)$, meaning external-sampling MCCFR requires asymptotically less time to compute an approximate equilibrium than vanilla CFR (and consequently chance-sampling CFR, which is identical to vanilla CFR in the absence of chance nodes).

**Theorem 5** *For any $p \in (0,1]$, when using outcome-sampling MCCFR where $\forall z \in Z$ either $\pi^\sigma_{-i}(z) = 0$ or $q(z) \geq \delta > 0$ at every timestep, with probability $1 - p$, average overall regret is bounded by $R_i^T \leq \left(1 + \frac{\sqrt{2}}{\sqrt{p}}\right) \left(\frac{1}{\delta}\right) \Delta_{u,i} M_i \sqrt{|A_i|}/\sqrt{T}$*

The proofs for the theorems in this section can be found in the paper's supplemental material and as an appendix of the associated technical report [10]. The supplemental material also presents a slightly complicated, but general result for any member of the MCCFR family, from which the two theorems presented above are derived.

| Game | $|H|$ ($10^6$) | $|\mathcal{I}|$ ($10^3$) | $l$ | $M_1$ | $M_2$ | $t_{vc}$ | $t_{os}$ | $t_{es}$ |
|---|---|---|---|---|---|---|---|---|
| OCP | 22.4 | 2 | 5 | 45 | 32 | 28s | 46$\mu$s | 99$\mu$s |
| Goof | 98.3 | 3294 | 14 | 89884 | 89884 | 110s | 150$\mu$s | 150ms |
| LTTT | 70.4 | 16039 | 18 | 1333630 | 1236660 | 38s | 62$\mu$s | 70ms |
| PAM | 91.8 | 20 | 13 | 9541 | 2930 | 120s | 85$\mu$s | 28ms |

Table 1: Game properties. The value of $|H|$ is in millions and $|\mathcal{I}|$ in thousands, and $l = \max_{h \in H} |h|$. $t_{vc}$, $t_{os}$, and $t_{es}$ are the average wall-clock time per iteration[4] for vanilla CFR, outcome-sampling MCCFR, and external-sampling MCCFR.

## 5    Experimental Results

We evaluate the performance of MCCFR compared to vanilla CFR on four different games. Goof-spiel [11] is a bidding card game where players have a hand of cards numbered 1 to $N$, and take turns secretly bidding on the top point-valued card in a point card stack using cards in their hands. Our version is less informational: players only find out the result of each bid and not which cards were used to bid, and the player with the highest total points wins. We use $N = 7$ in our experiments. One-Card Poker [12] is a generalization of Kuhn Poker [13], we use a deck of size 500. Princess and Monster [14, Research Problem 12.4.1] is a pursuit-evasion game on a graph, neither player ever knowing the location of the other. In our experiments we use random starting positions, a 4-connected 3 by 3 grid graph, and a horizon of 13 steps. The payoff to the evader is the number of steps uncaptured. Latent Tic-Tac-Toe is a twist on the classic game where moves are not disclosed until after the opponent's next move, and lost if invalid at the time they are revealed. While all of these games have imperfect information and roughly of similar size, they are a diverse set of games, varying both in the degree (the ratio of the number of information sets to the number of histories) and nature (whether due to chance or opponent actions) of imperfect information. The left columns of Table 1 show various constants, including the number of histories, information sets, game length, and M-values, for each of these domains.

We used outcome-sampling MCCFR, external-sampling MCCFR, and vanilla CFR to compute an approximate equilibrium in each of the four games. For outcome-sampling MCCFR we used an epsilon-greedy sampling profile $\sigma'$. At each information set, we sample an action uniformly randomly with probability $\epsilon$ and according to the player's current strategy $\sigma^t$. Through experimentation we found that $\epsilon = 0.6$ worked well across all games; this is interesting because the regret bound suggests $\delta$ should be as large as possible. This implies that putting some bias on the most likely outcome to occur is helpful. With vanilla CFR we used to an implementational trick called pruning to dramatically reduce the work done per iteration. When updating one player's regrets, if the other player has no probability of reaching the current history, the entire subtree at that history can be pruned for the current iteration, with no effect on the resulting computation. We also used vanilla CFR without pruning to see the effects of pruning in our domains.

Figure 1 shows the results of all four algorithms on all four domains, plotting approximation quality as a function of the number of nodes of the game tree the algorithm touched while computing. Nodes touched is an implementation-independent measure of computation; however, the results are nearly identical if total wall-clock time is used instead. Since the algorithms take radically different amounts of time per iteration, this comparison directly answers if the sampling variants' lower cost per iteration outweighs the required increase in the number of iterations. Furthermore, for any fixed game (and degree of confidence that the bound holds), the algorithms' average overall regret is falling at the same rate, $O(1/\sqrt{T})$, meaning that only their short-term rather than asymptotic performance will differ.

The graphs show that the MCCFR variants often dramatically outperform vanilla CFR. For example, in Goofspiel, both MCCFR variants require only a few million nodes to reach $\epsilon_\sigma < 0.5$ where CFR takes 2.5 billion nodes, three orders of magnitude more. In fact, external-sampling, which has the tightest theoretical computation-time bound, outperformed CFR and by considerable margins (excepting LTTT) in all of the games. Note that pruning is key to vanilla CFR being at all practical in these games. For example, in Latent Tic-Tac-Toe the first iteration of CFR touches 142 million nodes, but later iterations touch as few as 5 million nodes. This is because pruning is not possible

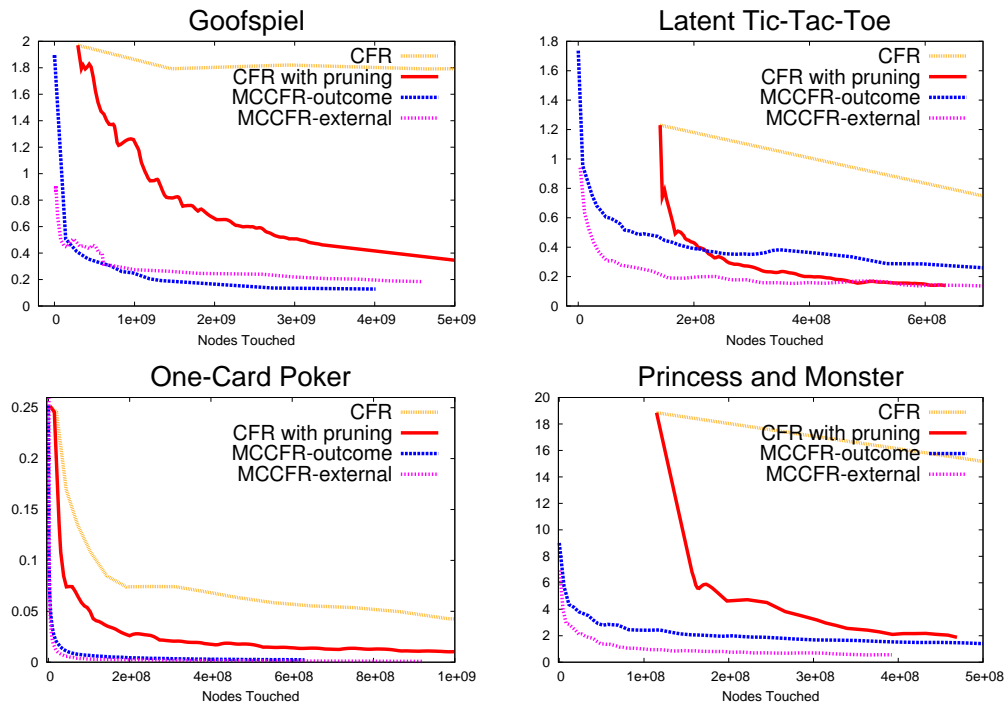

Figure 1: Convergence rates of Vanilla CFR, outcome-sampled MCCFR, and external-sampled MC-CFR for various games. The $y$ axis in each graph represents the exploitability of the strategies for the two players $\epsilon_\sigma$ (see Section 2.1).

in the first iteration. We believe this is due to dominated actions in the game. After one or two traversals, the players identify and eliminate dominated actions from their policies, allowing these subtrees to pruned. Finally, it is interesting to note that external-sampling was not uniformly the best choice, with outcome-sampling performing better in Goofspiel. With outcome-sampling performing worse than vanilla CFR in LTTT, this raises the question of what specific game properties might favor one algorithm over another and whether it might be possible to incorporate additional game specific constants into the bounds.

## 6 Conclusion

In this paper we defined a family of sample-based CFR algorithms for computing approximate equilibria in extensive games, subsuming all previous CFR variants. We also introduced two sampling schemes: outcome-sampling, which samples only a single history for each iteration, and external-sampling, which samples a deterministic strategy for the opponent and chance. In addition to presenting a tighter bound for vanilla CFR, we presented regret bounds for both sampling variants, which showed that external sampling with high probability gives an asymptotic computational time improvement over vanilla CFR. We then showed empirically in very different domains that the reduction in iteration time outweighs the increase in required iterations leading to faster convergence.

There are three interesting directions for future work. First, we would like to examine how the properties of the game effect the algorithms' convergence. Such an analysis could offer further algorithmic or theoretical improvements, as well as practical suggestions, such as how to choose a sampling policy in outcome-sampled MCCFR. Second, using outcome-sampled MCCFR as a general online regret minimizing technique in extensive games (when the opponents' strategy is not known or controlled) appears promising. It would be interesting to compare the approach, in terms of bounds, computation, and practical convergence, to Gordon's Lagrangian hedging [5]. Lastly, it seems like this work could be naturally extended to cases where we don't assume perfect recall. Imperfect recall could be used as a mechanism for abstraction over actions, where information sets are grouped by important partial sequences rather than their full sequences.

## Footnotes

[1]Traditionally, an information partition is not specified for chance. In fact, as long as the same chance information set cannot be revisited, it has no strategic effect on the game itself. However, this extension allows us to consider using the same sampled chance outcome for an entire set of histories, which is an important part of Zinkevich and colleagues' chance-sampling CFR variant.

[2]Regret-matching selects actions with probability proportional to their positive regret, *i.e.*,   $\sigma^t_i(a|I) = R^{T,+}_{i,\mathrm{imm}}(I, a)/\sum_{a' \in A(I)} R^{T,+}_{i,\mathrm{imm}}(I, a)$. Regret-matching satisfies Blackwell's approachability criteria. [7, 8]

[4]As measured on an 8-core Intel Xeon 2.5 GHz machine running Linux x86_64 kernel 2.6.27.

## References

[1] Martin Zinkevich, Michael Johanson, Michael Bowling, and Carmelo Piccione. Regret minimization in games with incomplete information. In *Advances in Neural Information Processing Systems 20 (NIPS)*, 2008.

[2] Andrew Gilpin, Samid Hoda, Javier Peña, and Tuomas Sandholm. Gradient-based algorithms for finding Nash equilibria in extensive form games. In *3rd International Workshop on Internet and Network Economics (WINE'07)*, 2007.

[3] D. Koller, N. Megiddo, and B. von Stengel. Fast algorithms for finding randomized strategies in game trees. In *Proceedings of the 26th ACM Symposium on Theory of Computing (STOC '94)*, pages 750–759, 1994.

[4] Martin Zinkevich, Michael Johanson, Michael Bowling, and Carmelo Piccione. Regret minimization in game with incomplete information. Technical Report TR07-14, University of Alberta, 2007. `http://www.cs.ualberta.ca/research/techreports/2007/TR07-14.php`.

[5] Geoffrey J. Gordon. No-regret algorithms for online convex programs. In *In Neural Information Processing Systems 19*, 2007.

[6] Martin J. Osborne and Ariel Rubinstein. *A Course in Game Theory*. MIT Press, 1994.

[7] Sergiu Hart and Andreu Mas-Colell. A simple adaptive procedure leading to correlated equilibrium. *Econometrica*, 68(5):1127–1150, September 2000.

[8] D. Blackwell. An analog of the minimax theorem for vector payoffs. *Pacific Journal of Mathematics*, 6:1–8, 1956.

[9] Peter Auer, Nicolò Cesa-Bianchi, Yoav Freund, and Robert E. Schapire. Gambling in a rigged casino: The adversarial multi-arm bandit problem. In *36th Annual Symposium on Foundations of Computer Science*, pages 322–331, 1995.

[10] Marc Lanctot, Kevin Waugh, Martin Zinkevich, and Michael Bowling. Monte carlo sampling for regret minimization in extensive games. Technical Report TR09-15, University of Alberta, 2009. `http://www.cs.ualberta.ca/research/techreports/2009/TR09-15.php`.

[11] S. M. Ross. Goofspiel — the game of pure strategy. *Journal of Applied Probability*, 8(3):621–625, 1971.

[12] Geoffrey J. Gordon. No-regret algorithms for structured prediction problems. Technical Report CMU-CALD-05-112, Carnegie Mellon University, 2005.

[13] H. W. Kuhn. Simplified two-person poker. *Contributions to the Theory of Games*, 1:97–103, 1950.

[14] Rufus Isaacs. *Differential Games: A Mathematical Theory with Applications to Warfare and Pursuit, Control and Optimization*. John Wiley & Sons, 1965.

